# A Small World Threshold
# for Economic Network Formation

**Eyal Even-Dar**
Computer and Information Science
University of Pennsylvania
Philadelphia, PA 19104
evendar@seas.upenn.edu

**Michael Kearns**
Computer and Information Science
University of Pennsylvania
Philadelphia, PA 19104
mkearns@cis.upenn.edu

## Abstract

We introduce a game-theoretic model for network formation inspired by earlier stochastic models that mix localized and long-distance connectivity. In this model, players may purchase edges at distance $d$ at a cost of $d^\alpha$, and wish to minimize the sum of their edge purchases and their average distance to other players. In this model, we show there is a striking "small world" threshold phenomenon: in two dimensions, if $\alpha < 2$ then every Nash equilibrium results in a network of *constant* diameter (independent of network size), and if $\alpha > 2$ then every Nash equilibrium results in a network whose diameter grows as a root of the network size, and thus is unbounded. We contrast our results with those of Kleinberg [8] in a stochastic model, and empirically investigate the "navigability" of equilibrium networks. Our theoretical results all generalize to higher dimensions.

## 1 Introduction

Research over the last decade from fields as diverse as biology, sociology, economics and computer science has established the frequent empirical appearance of certain structural properties in naturally occurring networks. These properties include small diameter, local clustering of edges, and heavy-tailed degree distributions [11]. Not content to simply catalog such apparently "universal" properties, many researchers have proposed stochastic models of decentralized network formation that can explain their emergence. A typical such model is known as *preferential attachment* [3], in which arriving vertices are probabilistically more likely to form links to existing vertices with high degree; this generative process is known to form networks with power law degree distributions.

In parallel with these advances, economists and computer scientists have examined models in which networks are formed due to "rational" or game-theoretic forces rather than probabilistic ones. In such models networks are formed via the self-interested behavior of individuals who benefit from participation in the network [7]. Common examples include models in which a vertex or player can purchase edges, and would like to minimize their average shortest-path distance to all other vertices in the jointly formed network. A player's overall utility thus balances the desire to purchase few edges yet still be "well-connected" in the network. While stochastic models for network formation define a (possibly complex) distribution over possible networks, the game-theoretic models are typically equated with their (possibly complex) set of (Nash) *equilibrium* networks. It is also common to analyze the so-called *Price of Anarchy* [9] in such models, which measures how much worse an equilibrium network can be than some measure of social or centralized optimality [6, 2, 5, 1].

In this paper we introduce and give a rather sharp analysis of a network formation model of the game-theoretic variety, but which was inspired by a striking result of Kleinberg [8] in a stochastic model, and thus forms a bridge between these two lines of thought. In Kleinberg's stochastic model, the network formation process begins on an underlying substrate network that is highly regular —

for instance, a grid in two dimensions. This regular substrate is viewed as a coarse model of "local" connectivity, such as one's geographically close neighbors. The stochastic process then adds "long-distance" edges to the grid, in an attempt to model connections formed by travel, chance meetings, and so on. Kleinberg's model assumes that the probability that an edge connecting two vertices whose *grid* distance is $d$ is proportional to $1/d^\alpha$ for some $\alpha > 0$ — thus, longer-distance edges are less likely, but will still appear in significant numbers due to the long tail of the generating distribution. An interesting recent empirical study [4] of the migration patterns of dollar bills provides evidence for the validity of such a model. In a theoretical examination of the "six degrees of separation" or "small world" folklore first popularized by the pioneering empirical work of Travers and Milgram [10], Kleinberg proved that *only for $\alpha = 2$* will the resulting network be likely to support the routing of messages on short paths using a natural distributed algorithm. For larger values of $\alpha$ the network simply does not have short paths (small diameter), and for smaller values the diameter is quite small, but the long-distance edges cannot be exploited effectively from only local topological information.

Our model and result can be viewed as an "economic" contrast to Kleinberg's. We again begin with a regular substrate like the grid in two dimensions; these edges are viewed as being provided free of charge to the players or vertices. A vertex $u$ is then free to purchase an edge to a vertex $v$ at grid distance $d = \delta(u, v)$ at a cost of $d^\alpha$ for $\alpha > 0$. Thus, longer-distance edges now have higher cost rather than lower probability, but again in a power law form. We analyze the networks that are Nash equilibria of a game in which each player's payoff is the negative of the sum of their edge purchases and average distances to the other vertices.

Our main result is a precise analysis of the diameter (longest shortest path between any pair of vertices) of equilibrium networks in this model. In particular, we show a sharp threshold result: *for any $\alpha < 2$*, every pure Nash equilibrium network has only *constant* diameter (that is, diameter independent of the network size $n$); and *for any $\alpha > 2$*, every pure Nash equilibrium has diameter that grows as a *root of the network size* (that is, unbounded and growing rapidly with $n$). In the full version, we show in addition that the threshold phenomenon occurs in mixed Nash equilibrium as well.

Despite the outward similarity, there are some important differences between our results and Kleinberg's. In addition to the proofs being essentially unrelated (since one requires a stochastic and the other an equilibrium analysis), Kleinberg's result establishes a "knife's edge" (fast routing only at $\alpha$ exactly 2), while ours is a threshold or phase transition — there is a broad range of $\alpha$ values yielding constant diameter, which sharply crosses over to polynomial growth at $\alpha = 2$. On the other hand, for $\alpha = 2$ Kleinberg establishes that in his model not only that there is small (though order $\log(n)^2$ rather than constant) diameter, but that short paths can be *navigated* by a naive greedy routing algorithm. However, simulation results discussed in Section 5 suggest that the equilibrium networks of our model do support fast routing as well. Like Kleinberg's results, all of ours generalize to higher dimensions as well, with the threshold occurring at $\alpha = r$ in $r$-dimensional space.

The outline of the paper is as follows. In Section 2 we define our game-theoretic model and introduce the required equilibrium concepts. In Section 3 we provide the constant diameter upper bound for $r = 2$ when $\alpha < 2$, and also even better constants for $\alpha \leq 1$. Section 4 provides the diameter lower bound for $\alpha > 2$, while in Section 5 we explore greedy routing in equilibrium networks via simulation.

## 2 Preliminaries

We devote this section to a formal definition of the model. We assume that the players are located on a grid, so each player $v$ is uniquely identified with a grid point $(a, b)$, where $1 \leq a, b \leq \sqrt{n}$; thus the total number of players is $n$. The action of player $v_i$ is a vector $s_i \in \{0, 1\}^n$ indicating which edges to other players $v_i$ has purchased. We let $s = s_1 \times \cdots \times s_n$ be the joint action of all the players, $v_1, ..., v_n$. We also use $s_{-i}$ to denote the joint action of all players except player $v_i$.

*The Graph.* The joint action $s$ defines an undirected graph $G(s)$ as follows. The nodes of $G(s)$ are the players $V = \{v_1, \ldots, v_n\}$. An edge $(v_i, v_j)$ is bought by player $v_i$ if and only if $s_i(j) = 1$. Let $E_i(s_i) = \{(v_i, v_j) \mid s_i(j) = 1\}$ be the set of edges bought by player $v_i$ and let $E(s) = \cup_{i \in V} E_i(s_i)$. The graph induced by $s$ is $G(s) = (V, E(s))$.

*Distances and Costs.* The grid defines a natural distance $\delta$. Let $v_i$ be the player identified with the grid point $(a, b)$ and $v_{i'}$ with $(a', b')$; then their grid distance is $\delta(v_i, v_{i'}) = |a - a'| + |b - b'|$. Next we define a natural family of edge cost functions in which the cost of an edge is a function of the grid distance:

$$c(v_i, v_j) = \begin{cases} 0 & \delta(v_i, v_j) = 1 \\ a\delta(v_i, v_j)^\alpha & \text{otherwise} \end{cases}$$

where $a, \alpha > 0$ are parameters of the model. Thus, grid edges are free to the players, and longer edges have a cost polynomial in their grid distance.

*The Game.* We are now ready to define the formal network formation game we shall analyze. The overall cost function $c_i$ of player $v_i$ is defined as

$$c_i(s) = c_i(s_i, s_{-i}) = \sum_{e \in E_i(s_i)} c(e) + \sum_{j=1}^{n} \Delta_{G(s)}(v_i, v_j)$$

where $\Delta_{G(s)}(u, v)$ is the shortest distance between $u$ and $v$ in $G(s)$. Thus, in this game player $i$ wishes to minimize $c_i(s)$, which requires balancing edge costs and shortest paths. We emphasize that players benefit from edge purchases by other players, since shortest paths are measured with respect to the overall graph formed by all edges purchased. The graph diameter is defined as $\max_{i,j} \Delta_{G(s)}(v_i, v_j)$.

*Equilibrium Concepts.* A joint action $s = s_1 \times \cdots \times s_n$ is said to be a *Nash equilibrium* if for every player $i$ and any alternative action $\hat{s}_i \in \{0, 1\}^n$, we have $c_i(s_i, s_{-i}) \leq c_i(\hat{s}_i, s_{-i})$. If $s$ is a Nash equilibrium we say that its corresponding graph $G(s)$ is an equilibrium graph. A joint action $s = s_1 \times \cdots \times s_n$ is said to be *link stable* if for every player $i$ and any alternative action $\hat{s}_i \in \{0, 1\}^n$ that differs from $s_i$ in *exactly one coordinate* (i.e. one edge), we have $c_i(s_i, s_{-i}) \leq c_i(\hat{s}_i, s_{-i})$. If $s$ is link stable we say that its corresponding graph $G(s)$ is a stable graph. Note that an equilibrium graph implies a link stable graph. Link stability means that the graph is stable under single-edge unilateral deviations (as opposed to Nash, which permits arbitrary unilateral deviations), and is a private case of the pairwise stability given notion given in [7]. The popularity of the link stable notion is due to its simplicity and due to the fact that it is easily computable, as opposed to computing best responses which in similar problems is known to be NP-Hard [6]. Note that as the grid edges are free, the diameter of an equilibrium or link stable graph is bounded by $2\sqrt{n}$.

## 3  Constant Diameter at Equilibrium for $\alpha \in [0, 2]$

In this section we analyze the diameter of equilibrium networks when $\alpha \in [0, 2]$. Our results actually hold under the more general notion of link stability as well. The following is the first of our two main theorems.

**Theorem 3.1** *For any constant $\epsilon > 0$, if $\alpha = 2 - \epsilon$, then there exists a constant $c(\alpha)$ such that for any $n$, all Nash equilibria or link stable graphs over $n$ players have diameter at most $c(\alpha)$.*

The proof of this theorem has a number of technical subtleties, so we first provide its intuition, which is illustrated in Figure 1(B). We analyze an equilibrium (or link stable) graph in stages, and focus on the distance of vertices to some focal player $u$. In each stage we argue that more grid-distant players have an incentive to purchase an edge to $u$ due to the centrality of $u$ in the graph.

We start with the following simple fact: for every nodes $v$ and $w$ we have that if $\delta(v, w) \leq d$ then $\Delta_{G(s)}(v, w) \leq d$ since all grid edges are free. We would like to show that even a stronger property holds — namely, that if $\delta(v, w) \leq d^\alpha$ then $\Delta_{G(s)}(w, v) \leq d$ for some $\alpha > 1$. Since this property is no longer simply implied by the grid edges, it requires arguing that grid-distant vertices have an incentive to purchase edges to each other. Suppose there are nodes $u$ and $v$ such that $\Delta_{G(s)}(u, v) \geq d$. We first define a "close" graph neighborhood of $u$, $S_u = \{w | \Delta_{G(s)}(u, w) \leq d/3\}$. Note that for every $w \in S_u$ we have that $\Delta_{G(s)}(v, w) \geq 2d/3$. Next we would like to claim that the cardinality of $S_u$ is large — thus $u$'s neighborhood is densely populated. For this we define $S_u^\delta = \{w | \delta(u, w) \leq d/3\} \subseteq S_u$. Using the grid topology (see Figure 1(A)) we see that $|S_u^\delta|$ is of order $d^2$.

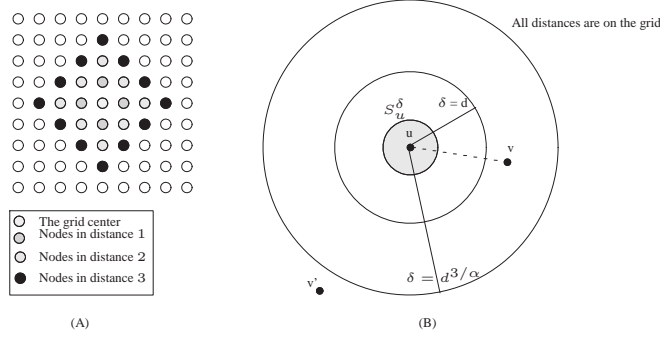

Figure 1: (A) The number of nodes at exact distance $k$ is exactly $4k$, while the number within $k$ is order $k^2$. (B) Illustration of the main argument of Theorem 3.1. Here $u$ and $v$ are vertices at grid distance $d$, while $u$ and $v'$ are at grid distance $d'$, where $d \leq d^{3/\alpha} \leq d'$. In the proof we use the size of $S_u^\delta$ to show that $v$ benefits by purchasing an edge to $u$ and thus must be distance 1 to $u$ in the equilibrium graph; this in turn allows us to argue that $v'$ wishes to purchase an edge to $u$ as well.

Now consider the benefit to $v$ of buying the edge $(v, u)$ (which is not in the graph since $\Delta_{G(s)}(u, v) \geq d$). Since the distance from $v$ to every node in $S_u$ is reduced by at least $d/3$ and the set size is at least order $d^2$, we have that the benefit is of order $d^3$. The fact that this edge was *not* bought implies that $\delta(v, u)^\alpha = \Omega(d^3)$. Therefore, we have that $\Delta_{G(s)}(u, v) \geq d$ implies that $\delta(u, v) = \Omega(d^{3/\alpha})$, which is the contrapositive of $\delta(u, v) = O(d^{3/\alpha})$ implies $\Delta_{G(s)}(u, v) \leq d$.

In other words, for "small enough" values of $\alpha$ (quantified in the full proof), vertices quite distant from $u$ in the grid have an incentive to buy an edge to $u$, by virtue of the dense population in $S_u^\delta$. But this in turn argues that the size of $S_u$ is even larger than $S_u^\delta$; we then "bootstrap" this argument to show that yet further vertices have an incentive to connect to $u$, and so on. We now proceed with the formal proof based on this argument.

**Lemma 3.2** *Let $G(s)$ be an equilibrium or link stable graph and $u$ be the grid center. Suppose that for every node $v$ such that $\delta(u, v) \leq d^\beta$ (where $\beta \geq 1$ and $d^\beta < \sqrt{n}/2$), we have that $\Delta_{G(s)}(u, v) \leq d$. Then for every $d$, and for every node $v$ such that $\delta(u, v) \leq 2^{1/\alpha}(d/3)^{\beta'}$, where $\beta' = \frac{2\beta+1}{\alpha}$, we have that $\Delta_{G(s)}(u, v) \leq d$.*

**Proof:** Let $v$ be a node such that $\Delta_{G(s)}(u, v) = d$ and let $S_u = \{w | \Delta_{G(s)}(u, w) \leq d/3\}$; observe that $d' = \min_{w \in S_u} \Delta_{G(s)}(w, v)$ is at least $\frac{2d}{3}$ and thus $v$'s benefit of buying the edge $(v, u)$ is at least $\frac{d}{3}|S_u|$. Next we would like to bound the size of $S_u$ from below. Using the topology of the grid, the grid the center node has $4k$ nodes (See Figure 1(A)) in exact grid distance $k$ (if $k \leq n/2$), which implies that the center node has $2k^2$ nodes in grid distance at most $k$. The set $S_u$ contains all nodes such that $\Delta_{G(s)}(u, w) \leq d/3$ by definition which implies by our assumption that it includes all nodes $w$ such that $\delta(u, w) \leq (d/3)^\beta$. Therefore, the size of $S_u$ is at least $2(d/3)^{2\beta}$. Now since $G(s)$ is an equilibrium or link stable graph, it means that $v$ would not like to buy the edge $(u, v)$ and thus

$$\delta(u, v)^\alpha > 2(d/3)^{2\beta} \cdot d/3 = 2\frac{d^{2\beta+1}}{3^{2\beta+1}}$$

Taking the $\alpha$ root, we have that $\Delta_{G(s)}(u, v) > d$ implies $\delta(u, v) \geq 2^{1/\alpha}\frac{d^{(2\beta+1)/\alpha}}{3^{(2\beta+1)/\alpha}}$, which is the contrapositive of $\delta(u, v) \leq 2^{1/\alpha}\frac{d^{(2\beta+1)/\alpha}}{3^{(2\beta+1)/\alpha}}$ implies $\Delta_{G(s)}(u, v) \leq d$, as required. ∎

Equipped with this lemma we can prove rather strong results regarding the case where $\alpha = 2 - \epsilon$, for $\epsilon > 0$. In the previous lemma there are two parts in the change of the radius — one is the exponent, which grows, and the second is that instead of having $d$ in the base we have only $d/3$. The next lemma shows that as long as $d$ is large enough we can ignore the fact that the base decreases from $d$ to $d/3$ — and thus "amplify" the exponent $\beta$ in the preceding analysis to a larger exponent $(1+\epsilon_1)\beta$.

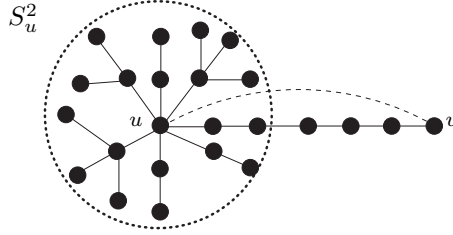

$S_u^2$

$u$

$v$

Figure 2: A graph with diameter of 6.

**Lemma 3.3** *(Amplification Lemma) Let $G(s)$ be an equilibrium or link stable graph. Let $\alpha = 2 - \epsilon$ for some $\epsilon > 0$. Let $c(\alpha)$ be a constant determined by subsequent analysis. Suppose that for every $d > c(\alpha)$, for every node $v$ such that $\delta(u, v) \le d^\beta$ (where $\beta \ge 1$, $d^\beta < \sqrt{n}/2$, and $u$ is the grid center), we have that $\Delta_{G(s)}(u, v) \le d$. Then for every $d > c(\alpha)$, for every node $v$ such that $\delta(u, v) \le d^{\beta'}$, where $\beta' = \beta(1 + \epsilon_1)$, we have that $\Delta_{G(s)}(u, v) \le d$, where $\epsilon_1 = \frac{\epsilon}{2(2-\epsilon)}$.*

**Proof:** Set $c(\alpha) = 3^{\frac{1+2\epsilon_1}{\epsilon_1}}$. By Lemma 3.2 we have that for every $d > 3^{\frac{1+2\epsilon_1}{\epsilon_1}}$ for every nodes $u$ and $v$ such that $\delta(u, v) \le \frac{(d/3)^{\hat{\beta}}}{2}$, where $\hat{\beta} = \frac{2\beta+1}{\alpha}$, we have that $\Delta_{G(s)}(u, v) \le d$.

$$\frac{(d/3)^{\hat{\beta}}}{2} = \frac{(d/3)^{\frac{2\beta+1}{\alpha}}}{2} = \frac{d^{(1+\frac{\epsilon}{2-\epsilon})\beta+1/\alpha}}{2 \cdot 3^{(1+\frac{\epsilon}{2-\epsilon})\beta+1/\alpha}} > \frac{d^{(1+\frac{\epsilon}{2-\epsilon})\beta}}{3^{(1+\frac{\epsilon}{2-\epsilon})\beta}}$$

$$= \frac{d^{(1+\epsilon_1)\beta}d^{\epsilon_1\beta}}{3^{(1+2\epsilon_1)\beta}} \ge d^{(1+\epsilon_1)\beta}$$

where both inequalities hold for $d \ge c(\alpha)$ ∎

Now we are ready to prove the main theorem of this section.

**Proof:** (Theorem 3.1) Let $c'(\alpha) = 3^{\frac{1+2\epsilon_1}{\epsilon_1}}$, where $\epsilon_1 = \frac{\epsilon}{2(2-\epsilon)}$ and let $u$ be the grid center. For every node $v$ such that $\delta(u, v) \le c'(\alpha)$, we must have $\Delta_{G(s)}(u, v) \le c'(\alpha)$, since all grid edges are part of $G(s)$. Next we prove that all nodes within grid distance $\sqrt{n}/2$ are within graph distance $c'(\alpha)$. Since $\Delta_{G(s)}(u, v) \le \delta(u, v)$, we can apply Lemma 3.3 to obtain that in radius $c'(\alpha)$ of $u$ in $G$, are all nodes $v$ such that $\delta(u, v) \le c'(\alpha)^{1+\epsilon_1}$. We repeat this argument recursively and obtain after the $k$-th time, that all nodes $v$ such that $\delta(u, v) \le c'(\alpha)^{(1+\epsilon_1)^k}$ satisfy $\Delta_{G(s)}(u, v) \le c'(\alpha)$. Taking $k = \log_{1+\epsilon_1}(\sqrt{n}/2)$, this implies that there are $n/2$ nodes within $c'(\alpha)$ from $u$. Now suppose there exists a node $v$ such that $\Delta_{G(s)}(u, v) \ge 3c'(\alpha)$. Then by buying the edge $(v, u)$, $u$'s benefit is at least $2c'(\alpha)n/2$ (we know that there are at least $n/2$ nodes within graph distance of $c'(\alpha)$ from $u$), while its cost is bounded by $\sqrt{n}^\alpha < n$ (since any node grid distance from $u$ is at most $\sqrt{n}$). Setting $c(\alpha) = 6c'(\alpha)$, we obtain the theorem. ∎

### 3.1 Even Smaller Constant Diameter at Equilibrium for $\alpha \le 1$

The constant diameter bound $c(\alpha)$ in Theorem 3.1 blows up as $\epsilon$ approaches 0. In this section we show that for $\alpha < 1$, rather small constant bounds hold. Note that when $\alpha \le 1$, the most expensive edge cost is bounded by $2a\sqrt{n}$, where $a$ is the edge cost constant. We will use this fact to show that every equilibrium graph $G(s)$ has a small constant diameter.

Let $u, v \in V$, we let $T_{G(s)}(u, v)$ be the set of all nodes that $u$ can reach on a shortest path that includes $v$. Formally, $T_{G(s)}(u, v) = \{w \mid \Delta_{G(s)}(u, w) = \Delta_{G(s)}(u, v) + \Delta_{G(s)}(v, w)\}$. We start by providing a technical lemma.

**Lemma 3.4** *Let $G(s)$ be an equilibrium or link stable graph. Let $u, v \in V$ be an arbitrary pair of players. If $(u, v) \notin E(s)$ then $\mid T_{G(s)}(u, v) \mid \le \frac{\delta(u,v)^\alpha}{\Delta_{G(s)}(u,v)-1}$.*

**Proof:** Buying the edge $(u, v)$ (at a cost of $\delta(u, v)^\alpha$) makes the distance from $u$ to every $w \in T_{G(s)}(u, v)$ shorter by $\Delta_{G(s)}(u, v) - 1$. However, $s$ is a Nash equilibrium, thus we know that the

edge $(u, v)$ was not bought. This implies that the benefit $(\Delta_{G(s)}(u, v) - 1) \cdot |T_{G(s)}(u, v)|$ from buying the edge is bounded by $\delta(u, v)^\alpha$. ∎

**Lemma 3.5** *Let $G(s) = (V, E(s))$ be an equilibrium graph and let $u, v \in V$.*

- *If $\alpha < 1$ then $\Delta_{G(s)}(u, v) \leq 5$.*

- *If $\alpha = 1$ then $\Delta_{G(s)}(u, v) \leq 2\lceil a^2 + 4 \rceil$*

**Proof:** We prove for the case that the cost functions is $a\delta(u, v)$ and omit the proof for the case where $\alpha < 1$ which is similar. Assume for contradiction that there exist a node $v$ such that $\Delta_{G(s)}(u, v) \geq \lceil a^2 + 4 \rceil + 1$, where $u$ is the grid center node (note that the grid distance from $u$ is bounded by $\sqrt{n}$). Let $S_u^2 = \{w | \Delta_{G(s)}(u, w) \leq 2\}$ be the set of nodes at a distance of at most 2 from $u$ (See Figure 3.1) including $u$. We first bound the size of $S_u^2$. For every node $w \in S_u^2$ we have $\Delta_{G(s)}(u, w) \geq \lceil a^2 + 4 \rceil - 1$. Buying the edge $(v, u)$ makes the distance between $v$ and every $w \in S_u^2$ at most 3. Thus, the benefit from buying the edge $(v, u)$ is at least $(\lceil a^2 + 4 \rceil - 1 - 3)|S_u^2| = \lceil a^2 \rceil |S_u^2|$. However, the edge $(v, u) \notin E(s)$ and is not part of the equilibrium graph. Therefore, the benefit from buying it is at most $\delta(v, u)$. This implies that $\lceil a^2 \rceil |S_u^2| \leq \delta(v, u) \leq a\sqrt{n}$. Now we look on a shortest paths tree rooted at $u$. There are at most $\sqrt{n}/\lceil a^2 \rceil - 2$ nodes at a distance of 2 from $u$. Each one of them has at most $a\sqrt{n}$ descendants by Lemma 3.4. Since the graph is connected, we get that $a\sqrt{n}/\lceil a^2 \rceil (a\sqrt{n} - 2) + a\sqrt{n}/\lceil a^2 \rceil \geq n$, which is a contradiction. ∎

## 3.2 The Case $\alpha = 2$

In this case we obtain neither a constant upper bound nor a polynomial lower bound. We show that for $\alpha = 2$ the diameter is bounded by $O(\sqrt{n}^{2/\sqrt{\log n}})$, which is bounded by $\sqrt{n}^c$ for every constant $c$ (i.e. this bound is very small as well); however it bounds from above any polylogarithmic function.

**Theorem 3.6** *Let the edge cost be $c((u, v)) = \delta(u, v)^2$, and let $G(s) = (V, E(s))$ be an equilibrium or link stable graph . Then the graph diameter is bounded by $O(\sqrt{n}^{2/\sqrt{\log n}})$.*

**Proof:** We again apply Lemma 3.2 repeatedly, but now with $\alpha = 2$. After applying it for the first time we have that all nodes which are in grid distance $(\frac{d}{3})^{3/2}$ from $u$ the grid center are within graph distance $d$. Recall that $S_u = \{w | \Delta_{G(s)}(u, w) \leq d/3\}$. Using the same arguments we construct a series of distances $x_k$, such that if $\delta(u, v) \leq x_k$ then $\Delta_{G(s)}(u, v) \leq d$. We begin with $x_1$ as $(\frac{d}{3})^{3/2}$ and now compute $x_2$:

$$x_2^2 > \frac{d}{3}|S_u| = \frac{d}{3}(\frac{d^{3/2}}{3^{3/2}}/3^{3/2})^2$$

Solving it we obtain that $x_2 = \frac{d^{4/2}}{3^{7/2}}$. Suppose that after repeating the argument for the $k$th time we have that $x_k$ is at least $d^{a_k}/3^{b_k}$. Using this bound we derive a lower bound on the size of $S_u$ and obtain the following bound for the $k + 1$ iteration:

$$x_{k+1}^2 > \frac{d}{3}|S_u| = \frac{d}{3}(\frac{d^{a_k}/3^{a_k}}{3^{b_k}})^2$$

Thus we obtain that $x_{k+1} = \frac{d^{a_k+1/2}}{3^{b_k+a_k+1/2}}$ and

$$a_{k+1} = a_k + 1/2$$
$$b_{k+1} = b_k + a_k + 1/2.$$

Our next goal is to estimate $a_k$ and $b_k$. The estimation of $a_k$ is straight forward and $a_k = k/2 + 1$. For $b_k$ it is enough for our needs to consider an upper bound; since we have $b_{k+1} = b_k + k/2 + 3/2$, one can easily verify that $k^2/2$ is an upper bound for $k \geq 3$. Therefore, in order to provide an upper bound on the distance form the center grid $u$ we would like to find an initial $d$ such that

$$\exists k \text{ such that } \frac{d^{k/2}}{3^{k^2/2}} \geq \sqrt{n}$$

and $d$ is minimal. This clearly holds for $d = n^{2/\sqrt{\log n}}$ and $k = \sqrt{\log n}$ and can be shown to be the minimal value for which it holds. Now using similar arguments to previous proofs we show that every other node cannot be further away from $u$. ∎

# 4 Polynomial Diameter at Equilibrium for $\alpha > 2$

We now give our second main result, which states that for $\alpha > 2$ the diameter grows as a root of $n$ and is thus unbounded.

**Theorem 4.1** *For any $\alpha$, the diameter of any Nash equilibrium or link stable graph is $\Omega(\sqrt{n}^{\frac{\alpha-2}{\alpha+1}})$.*

Before giving the proof we note that this bound implies a trivial lower bound of a constant for $\alpha \leq 2$, and a polynomial for $\alpha > 2$. For instance, setting $\alpha = 3$ we obtain a lower bound of $\Omega(\sqrt{n}^{1/4})$. We first provide a simple lemma (stated without proof) regarding the influence of one edge on a connected graph's diameter.

**Lemma 4.2** *Let $G = (V, E)$ be a connected graph with diameter $C$, and Let $G' = (V, E \bigcup \{e\})$ for any edge $e$ then the diameter of $G'$ is at least $C/2$.*

**Proof:** (Theorem 4.1) Let $D$ be the diameter of an equilibrium graph, and $d$ be the grid distance of $(w, v)$ the most expensive edge bought in $G$, note that the most expensive edge corresponds to the longest edge in grid distance terms. First we observe that $D \geq 2\sqrt{n}/d$, as the grid diameter is $2\sqrt{n}$ and the fastest way to traverse it is through edges of maximal length which is $d$. By Lemma 4.2 the benefit of buying an edge $(u, v)$ is at most $2D(n-3)$, since the diameter before was at most $2D$ and the distance to your two neighbor and yourself has not been changed. Therefore, have $\delta(u, v)^\alpha = d^\alpha \leq 2D(n-3)$. Next we use the two simple bounds

$$d^\alpha \leq 2Dn \tag{1}$$
$$2\sqrt{n}/d \leq D \tag{2}$$

Substituting the bound of $d$ in Equation 2 into equation 1 we obtain that

$$(2\sqrt{n}/D)^\alpha \leq 2Dn$$
$$\frac{(2\sqrt{n})^\alpha}{2n} \leq D^{1+\alpha}$$
$$c(\sqrt{n}^{\frac{\alpha-2}{1+\alpha}}) \leq D$$

as required. ∎

# 5 Simulations

The analyses we have considered so far examine static properties of equilibrium and link stable graphs, and as such do not shed light on natural dynamics that might lead to them. In this section we briefly describe dynamical simulations on a $100 \times 100$ grid (which has $10^8$ possible edges). At each iteration a random vertex $u$ is selected. With probability $1/2$, an existing edge of $u$ (grid or long-distance) is selected at random, and we compute whether (given the current global configuration of the graph), $u$ would prefer *not* to purchase this edge, in which case it is deleted. With probability $1/2$, we instead select a second random vertex $v$, and compute whether (again given the global graph) $u$ would like to purchase the edge $(u, v)$, in which case it is added. Note that if this dynamic converges, it is to a link stable graph and not necessarily a Nash equilibrium, since only single-edge deviations are considered.

The left panel of Figure 3 shows the worst-case diameter as a function of the number of iterations, and demonstrates the qualitative validity of our theory for this dynamic. For $\alpha = 1, 2$ the diameter quickly falls to a rather small value (less than 10). The asymptotes for $\alpha = 3, 4$ are considerably higher.

The right panel revisits the question that was the primary interest of Kleinberg's work [8], namely the efficiency of "naive" or greedy navigation or routing. If we wish to route a message from the grid center to a randomly chosen destination, and the message is always forwarded from its current vertex to the graph neighbor whose grid address is closest to the destination, how long will it take? Kleinberg was the first to observe and explain the fact that the mere existence of short paths (small diameter) may not be sufficient for such greedy local routing algorithms to *find* the short paths. In the

right panel of Figure 3 we show that the routing efficiency does in fact seem to echo our theoretical results — for the aforementioned dynamic, very short paths (only slightly higher than the diameter) are found for small $\alpha$, much longer paths for larger $\alpha$.

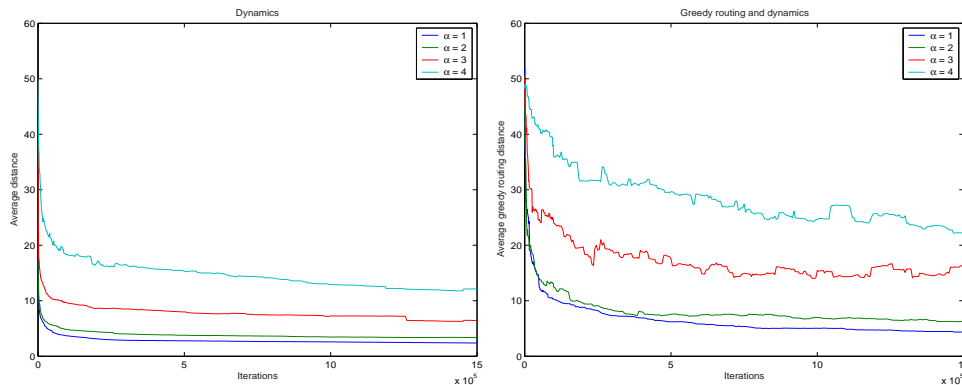

Figure 3: Left panel: graph diameter vs. iterations for a simple dynamic. Right panel: greedy routing efficiency vs. iterations for the same dynamic.

## 6   Extensions

We conclude by briefly mentioning generalizations of our theoretical results that we omit detailing. All of the results carry over higher dimensions, where the threshold phenomenon takes place at $\alpha$ equaling the grid dimension. We can also easily handle the case where the grid wraps around rather than having boundaries. We can also generalize to the pairwise link stability notion of [7], in which that the cost of each link is shared between the end points of the edge. Finally, we can construct network that are stable.

## References

[1] S. Albers, S. Eilts, E. Even-Dar, Y. Mansour, and L. Roditty. On Nash equilibria for a network creation game. In *Proc. of SODA*, pages 89–98, 2006.

[2] E. Anshelevich, A. Dasgupta, J. Kleinberg, E. Tardos, T. Wexler, and T. Roughgarden. The price of stability for network design with fair cost allocation. In *Proc. of FOCS*, pages 295–304, 2004.

[3] Albert-László Barabási and R. Albert. Emergence of scaling in random networks. *Science*, 286:509–512, 1999.

[4] D. Brockmann, L. Hufnagel, and T. Geisel. The scaling laws of human travel. *Nature*, 439:462–465, 2005.

[5] J. Corbo and D.C. Parkes. The price of selfish behavior in bilateral network formation. In *Proc. of PODC*, pages 99–107, 2005.

[6] A. Fabrikant, A. Luthra, E. Maneva, C.H̃. Papadimitriou, and S. Shenker. On a network creation game. In *Proc. of PODC*, pages 347–351, 2003.

[7] M. Jackson. A survey of models of network formation:stability and efficiency. In G. Demange and M. Wooders, editors, *Group Formation in Economics: Networks, Clubs and Coalitions*. 2003.

[8] Jon Klienberg. Navigation in a small world. *Nature*, 406:845, 2000.

[9] E. Koutsoupias and C. H. Papadimitriou. Worst-case equilibria. In *Proceedings of 16th STACS*, pages 404–413, 1999.

[10] J. Travers and S. Miligram. An expiermental study of small world problem. *Sociometry*, 32:425, 1969.

[11] Duncan J. Watts. *Six Degrees: The Science of a Connected Age*. W. W. Norton, Cambridge, Mass., 2003.
